# High-Dimensional Graphical Model Selection: Tractable Graph Families and Necessary Conditions

**Anima Anandkumar**
Dept. of EECS,
Univ. of California
Irvine, CA, 92697
a.anandkumar@uci.edu

**Vincent Y.F. Tan**
Dept. of ECE,
Univ. of Wisconsin
Madison, WI, 53706.
vtan@wisc.edu

**Alan S. Willsky**
Dept. of EECS
Massachusetts Inst. of Technology,
Cambridge, MA, 02139.
willsky@mit.edu

## Abstract

We consider the problem of Ising and Gaussian graphical model selection given $n$ i.i.d. samples from the model. We propose an efficient threshold-based algorithm for structure estimation based on conditional mutual information thresholding. This simple local algorithm requires only low-order statistics of the data and decides whether two nodes are neighbors in the unknown graph. We identify graph families for which the proposed algorithm has low sample and computational complexities. Under some transparent assumptions, we establish that the proposed algorithm is structurally consistent (or sparsistent) when the number of samples scales as $n = \Omega(J_{\min}^{-4} \log p)$, where $p$ is the number of nodes and $J_{\min}$ is the minimum edge potential. We also develop novel non-asymptotic techniques for obtaining necessary conditions for graphical model selection.

**Keywords:** Graphical model selection, high-dimensional learning, local-separation property, necessary conditions, typical sets, Fano's inequality.

## 1 Introduction

The formalism of probabilistic graphical models can be employed to represent dependencies among a large set of random variables in the form of a graph [1]. An important challenge in the study of graphical models is to learn the unknown graph using samples drawn from the graphical model. The general structure estimation problem is NP-hard [2]. In the *high-dimensional* regime, structure estimation is even more difficult since the number of available observations is typically much smaller than the number of dimensions (or variables). One of the goals is to characterize tractable model classes for which consistent graphical model selection can be guaranteed with low computational and sample complexities.

The seminal work by Chow and Liu [3] proposed an efficient algorithm for maximum-likelihood structure estimation in tree-structured graphical models by reducing the problem to a maximum weight spanning tree problem. A more recent approach for efficient structure estimation is based on convex-relaxation [4–6]. The success of such methods typically requires certain "incoherence" conditions to hold. However, these conditions are NP-hard to verify for general graphical models.

We adopt an alternative paradigm in this paper and instead analyze a simple local algorithm which requires only low-order statistics of the data and makes decisions on whether two nodes are neighbors in the unknown graph. We characterize the class of Ising and Gaussian graphical models for which we can guarantee efficient and consistent structure estimation using this simple algorithm. The class of graphs is based on a *local-separation* property and includes many well-known random graph families, including *locally-tree* like graphs such as large girth graphs, the Erdős-Rényi random graphs [7] and power-law graphs [8], as well as graphs with short cycles such as bounded-degree graphs, and small-world graphs [9]. These graphs are especially relevant in modeling social networks [10, 11].

### 1.1 Summary of Results

We propose an algorithm for structure estimation, termed as conditional mutual information thresholding (CMIT), which computes the minimum empirical conditional mutual information of a given node pair over conditioning sets of bounded cardinality $\eta$. If the minimum exceeds a given threshold (depending on the number of samples $n$ and the number of nodes $p$), the node pair is declared as an edge. This test has a low computational complexity of $O(p^{\eta+2})$ and requires only low-order statistics (up to order $\eta + 2$) when $\eta$ is small. The parameter $\eta$ is an upper bound on the size of local vertex-separators in the graph, and is small for many common graph families, as discussed earlier. We establish that under a set of mild and transparent assumptions, structure learning is consistent in high-dimensions for CMIT when the number of samples scales as $n = \Omega(J_{\min}^{-4} \log p)$, for a $p$-node graph, where $J_{\min}$ is the minimum (absolute) edge-potential in the model.

We also develop novel techniques to obtain necessary conditions for consistent structure estimation of Erdős-Rényi random graphs. We obtain non-asymptotic bounds on the number of samples $n$ in terms of the expected degree and the number of nodes of the model. The techniques employed are information-theoretic in nature and combine the use of Fano's inequality and the so-called asymptotic equipartition property.

Our results have many ramifications: we explicitly characterize the tradeoff between various graph parameters such as the maximum degree, girth and the strength of edge potentials for efficient and consistent structure estimation. We draw connections between structure learning and the statistical physical properties of the model: learning is fundamentally related to the absence of long-range dependencies in the model, i.e., the regime of *correlation decay*. The notion of correlation decay on Ising models has been extensively characterized [12], but its connections to structure learning have only been explored in a few recent works (e.g., [13]). This work establishes that consistent structure learning is feasible under a slightly weaker condition than the usual notion of correlation decay for a rich class of graphs. Moreover, we show that the Gaussian analog of correlation decay is the so-called *walk-summability* condition [14]. This is a somewhat unexpected and surprising connection since walk-summability is a condition to characterize the performance of inference algorithms such as loopy belief propagation (LBP). Our work demonstrates that both successful inference and learning hinge on similar properties of the Gaussian graphical model.

## 2 Preliminaries

### 2.1 Graphical Models

A $p$-dimensional *graphical model* is a family of $p$-dimensional multivariate distributions Markov on some undirected graph $G_=(V, E)$ [1]. Each node in the graph $i \in V$ is associated to a random variable $X_i$ taking values in a set $\mathcal{X}$. We consider both discrete (in particular Ising) models where $\mathcal{X}$ is a finite set and Gaussian models where $\mathcal{X} = \mathbb{R}$. The set of edges $E$ captures the set of conditional-independence relationships among the random variables. More specifically, the vector of random variables $\mathbf{X} := (X_1, \ldots, X_p)$ with joint distribution $P$ satisfies the *global Markov* property with respect to a graph $G$, if for all disjoint sets $A, B \subset V$, we have

$$P(\mathbf{x}_A, \mathbf{x}_B | \mathbf{x}_S) = P(\mathbf{x}_A | \mathbf{x}_S) P(\mathbf{x}_B | \mathbf{x}_S). \tag{1}$$

where set $S$ is a *separator*[1]between $A$ and $B$. The Hammersley-Clifford theorem states that under the *positivity* condition, given by $P(\mathbf{x}) > 0$ for all $\mathbf{x} \in \mathcal{X}^p$ [1], the model $P$ satisfies the global Markov property according to a graph $G$ if and only if it factorizes according to the cliques of $G$.

We consider the class of Ising models, i.e., binary pairwise models which factorize according to the edges of the graph. More precisely, the probability mass function (pmf) of an Ising model is

$$P(\mathbf{x}) \propto \exp\left[\frac{1}{2}\mathbf{x}^T \mathbf{J}_G \mathbf{x} + \mathbf{h}^T \mathbf{x}\right], \quad \mathbf{x} \in \{-1, 1\}^p. \tag{2}$$

For Gaussian graphical models, the probability density function (pdf) is of the form,

$$f(\mathbf{x}) \propto \exp\left[-\frac{1}{2}\mathbf{x}^T \mathbf{J}_G \mathbf{x} + \mathbf{h}^T \mathbf{x}\right], \quad \mathbf{x} \in \mathbb{R}^p. \tag{3}$$

In both the cases, the matrix $\mathbf{J}_G$ is called the potential or information matrix and $\mathbf{h}$, the potential vector. For both Ising and Gaussian models, the sparsity pattern of the matrix $\mathbf{J}_G$ corresponds to that of the graph $G$, i.e., $J_G(i,j) = 0$ if and only if $(i,j) \notin G$.

We assume that the potentials are uniformly bounded above and below as:

$$J_{\min} \leq |J_G(i,j)| \leq J_{\max}, \quad \forall (i,j) \in G. \tag{4}$$

Our results on structure learning depend on $J_{\min}$ and $J_{\max}$, which is fairly natural – intuitively, models with edge potentials which are "too small" or "too large" are harder to learn than those with comparable potentials, i.e., homogenous models.

Notice that the conventional parameterizations for the Ising models in (2) and the Gaussian models in (3) are slightly different. Without loss of generality, for Ising model, we assume that $J(i,i) = 0$ for all $i \in V$. On the other hand, in the Gaussian setting, we assume that the diagonal elements of the inverse covariance (or information) matrix $\mathbf{J}_G$ are normalized to unity ($J(i,i) = 1, \ i \in V$), and that $\mathbf{J}_G$ can be decomposed as $\mathbf{J}_G = \mathbf{I} - \mathbf{R}_G$, where $\mathbf{R}_G$ is the matrix of partial correlation coefficients [14].

We consider the problem of *structure learning*, which involves the estimation of the edge set of the graph $G$ given $n$ i.i.d. samples $\mathbf{X}_1, \ldots, \mathbf{X}_n$ drawn either from the Ising model in (2) or the Gaussian model in (3). We consider the high-dimensional regime, where both $p$ and $n$ grow simultaneously; typically, the growth of $p$ is much faster than that of $n$.

## 2.2   Tractable Graph Families

We consider the class of graphical models Markov on a graph $G_p$ belonging to some ensemble $\mathcal{G}(p)$ of graphs with $p$ nodes. We emphasize that in our formulation the graph ensemble $\mathcal{G}(p)$ can either be deterministic or random – in the latter, we also specify a probability measure over the set of graphs in $\mathcal{G}(p)$. In the random setting, we use the term *almost every* (a.e.) graph $G \sim \mathcal{G}(p)$ satisfies a certain property $\mathcal{Q}$ (for example, connectedness) if $\lim_{p \to \infty} P[G_p \text{ satisfies } \mathcal{Q}] = 1$. In other words, the property $\mathcal{Q}$ holds asymptotically almost surely[2] (a.a.s.) with respect to the random graph ensemble $\mathcal{G}(p)$. Intuitively, this means that graphs that have a vanishing probability of occurrence as $p \to \infty$ are ignored. Our conditions and theoretical guarantees will be based on this notion for random graph ensembles.

We now characterize the ensemble of graphs amenable for consistent structure estimation. For $\gamma \in \mathbb{N}$, let $B_\gamma(i; G)$ denote the set of vertices within distance $\gamma$ from node $i$ with respect to graph $G$. Let $H_{\gamma,i} := G(B_\gamma(i; G))$ denote the subgraph of $G$ spanned by $B_\gamma(i; G)$, but in addition, we retain the nodes not in $B_\gamma(i; G)$ (and remove the corresponding edges).

**Definition 1 ($\gamma$-Local Separator)** *Given a graph $G$, a $\gamma$-local separator $S_\gamma(i,j)$ between $i$ and $j$, for $(i,j) \notin G$, is a minimal vertex separator[3] with respect to the subgraph $H_{\gamma,i}$. The parameter $\gamma$ is referred to as the* path threshold *for local separation.*

In other words, the $\gamma$-local separator $S_\gamma(i,j)$ separates nodes $i$ and $j$ with respect to paths in $G$ of length at most $\gamma$. We now characterize the ensemble of graphs based on the size of local separators.

**Definition 2 (($\eta, \gamma$)-Local Separation Property)** *An ensemble of graphs $\mathcal{G}(p; \eta, \gamma)$ satisfies $(\eta, \gamma)$-local separation property if for a.e. $G_p \in \mathcal{G}(p; \eta, \gamma)$,*

$$\max_{(i,j) \notin G_p} |S_\gamma(i,j)| \leq \eta. \tag{5}$$

In Section 3, we propose an efficient algorithm for graphical model selection when the underlying graph belongs to a graph ensemble $\mathcal{G}(p; \eta, \gamma)$ with sparse local node separators (i.e., with small $\eta$). Below we provide examples of three graph families which satisfy (5) for small $\eta$.

**(Example 1) Bounded Degree:** Any (deterministic or random) ensemble of degree-bounded graphs $\mathcal{G}_{\mathrm{Deg}}(p, \Delta)$ satisfies the $(\eta, \gamma)$-local separation property with $\eta = \Delta$ and every $\gamma \in \mathbb{N}$. Thus, our algorithm consistently recovers graphs with small (bounded) degrees ($\Delta = O(1)$). This case was considered previously in several works, e.g. [15, 16].

**(Example 2) Bounded Local Paths:** The $(\eta, \gamma)$-local separation property also holds when there are at most $\eta$ paths of length at most $\gamma$ in $G$ between any two nodes (henceforth, termed as the $(\eta, \gamma)$-*local paths property*). In other words, there are at most $\eta - 1$ overlapping[4] cycles of length smaller than $2\gamma$. Thus, a graph with girth $g$ (length of the shortest cycle) satisfies the $(\eta, \gamma)$-local separation property with $\eta = 1$ and $\gamma = g$. For example, the bipartite Ramanujan graph [17, p. 107] and the random Cayley graphs [18] have large girths. The girth condition can be weakened to allow for a small number of short cycles, while not allowing for overlapping cycles. Such graphs are termed as *locally tree-like*. For instance, the ensemble of Erdős-Rényi graphs $\mathcal{G}_{\mathrm{ER}}(p, c/p)$, where an edge between any node pair appears with a probability $c/p$, independent of other node pairs, is locally tree-like. It can be shown that $\mathcal{G}_{\mathrm{ER}}(p, c/p)$ satisfies $(\eta, \gamma)$-local separation property with $\eta = 2$ and $\gamma \leq \frac{\log p}{4 \log c}$ a.a.s. Similar observations apply for the more general *scale-free* or *power-law* graphs [8, 19]. Along similar lines, the ensemble of $\Delta$-random regular graphs, denoted by $\mathcal{G}_{\mathrm{Reg}}(p, \Delta)$, which is the uniform ensemble of regular graphs with degree $\Delta$ has no overlapping cycles of length at most $\Theta(\log_{\Delta - 1} p)$ a.a.s. [20, Lemma 1].

**(Example 3) Small-World Graphs:** The class of hybrid graphs or augmented graphs [8, Ch. 12] consist of graphs which are the union of two graphs: a "local" graph having short cycles and a "global" graph having small average distances between nodes. Since the hybrid graph is the union of these local and global graphs, it simultaneously has large degrees and short cycles. The simplest model $\mathcal{G}_{\mathrm{Watts}}(p, d, c/p)$, first studied by Watts and Strogatz [9], consists of the union of a $d$-dimensional grid and an Erdős-Rényi random graph with parameter $c$. One can check that a.e. graph $G \sim \mathcal{G}_{\mathrm{Watts}}(p, d, c/p)$ satisfies $(\eta, \gamma)$-local separation property in (5), with $\eta = d + 2$ and $\gamma \leq \frac{\log p}{4 \log c}$. Similar observations apply for more general hybrid graphs studied in [8, Ch. 12].

## 3 Method and Guarantees

### 3.1 Assumptions

(A1) **Scaling Requirements:** We consider the asymptotic setting where both the number of variables (nodes) $p$ and the number of samples $n$ go to infinity. We assume that the parameters $(n, p, J_{\min})$ scale in the following fashion:[5]

$$n = \omega(J_{\min}^{-4} \log p). \tag{6}$$

We require that the number of nodes $p \to \infty$ to exploit the local separation properties of the class of graphs under consideration.

(A2a) **Strict Walk-summability for Gaussian Models:** The Gaussian graphical model Markov on almost every $G_p \sim \mathcal{G}(p)$ is $\alpha$-walk summable, i.e.,

$$\|\overline{\mathbf{R}}_{G_p}\| \leq \alpha < 1, \tag{7}$$

where $\alpha$ is a constant (i.e., is not a function of $p$), $\overline{\mathbf{R}}_{G_p} := [|R_{G_p}(i, j)|]$ is the entry-wise absolute value of the partial correlation matrix $\mathbf{R}_{G_p}$. In addition, $\|\cdot\|$ denotes the spectral norm, which for symmetric matrices, is given by the maximum absolute eigenvalue.

(A2b) **Bounded Potentials for Ising Models:** The Ising model Markov on a.e. $G_p \sim \mathcal{G}(p)$ has its maximum absolute potential below a threshold $J^*$. More precisely,

$$\alpha := \frac{\tanh J_{\max}}{\tanh J^*} < 1. \tag{8}$$

Furthermore, the ratio $\alpha$ in (8) is not a function of $p$. See [21, 22] for an explicit characterization of $J^*$ for specific graph ensembles.

(A3) **Local-Separation Property:** We assume that the ensemble of graphs $\mathcal{G}(p; \eta, \gamma)$ satisfies the $(\eta, \gamma)$-local separation property with $\eta, \gamma \in \mathbb{N}$ satisfying:

$$\eta = O(1), \quad J_{\min} \alpha^{-\gamma} = \widetilde{\omega}(1), \tag{9}$$

where $\alpha$ is given by (7) for Gaussian models and by (8) for Ising models.[6] We can weaken the second requirement in (9) as $J_{\min}\alpha^{-\gamma} = \omega(1)$ for deterministic graph families (rather than random graph ensembles).

(A4) **Edge Potentials:** The edge potentials $\{J_{i,j}, (i,j) \in G\}$ of the Ising model are assumed to be generically drawn from $[-J_{\max}, -J_{\min}] \cup [J_{\min}, J_{\max}]$, i.e., our results hold except for a set of Lebesgue measure zero. We also characterize specific classes of models where this assumption can be removed and we allow for all choices of edge potentials. See [21, 22] for details.

The above assumptions are very general and hold for a rich class of models. Assumption (A1) stipulates the scaling requirements of number of samples for consistent structure estimation. Assumption (A2) and (A4) impose constraints on the model parameters. Assumption (A3) requires the local-separation property described in Section 2.2 with the path threshold $\gamma$ satisfying (9). We provide examples of graphs where the above assumptions are met.

**Gaussian Models on Girth-bounded Graphs:** Consider the ensemble of graphs $\mathcal{G}_{\mathrm{Deg,Girth}}(p; \Delta, g)$ with maximum degree $\Delta$ and girth $g$. We now derive a relationship between $\Delta$ and $g$, for the above assumptions to hold. It can be established that for the walk-summability condition in (A2a) to hold for Gaussian models, we require that $J_{\max} = O(1/\Delta)$. When the minimum edge potential achieves this bound ($J_{\min} = \Theta(1/\Delta)$), a sufficient condition for (A3) to hold is given by

$$\Delta\alpha^g = o(1). \tag{10}$$

In (10), we notice a natural tradeoff between the girth and the maximum degree of the graph ensemble for successful estimation under our framework: graphs with large degrees can be learned efficiently if their girths are large. Indeed, in the extreme case of trees which have infinite girth, in accordance with (10), there is no constraint on the node degrees for consistent graphical model selection and recall that the Chow-Liu algorithm [3] is an efficient method for model selection on tree-structured graphical models.

Note that the condition in (10) allows for the maximum degree bound $\Delta$ to grow with the number of nodes as long as the girth $g$ also grows appropriately. For example, if the maximum degree scales as $\Delta = O(\mathrm{poly}(\log p))$ and the girth scales as $g = O(\log \log p)$, then (10) is satisfied. This implies that graphs with fairly large degrees and short cycles can be recovered successfully consistently using the algorithm in Section 3.2.

**Gaussian Models on Erdős-Rényi and Small-World Graphs:** We can also conclude that a.e. Erdős-Rényi graph $G \sim \mathcal{G}_{\mathrm{ER}}(p, c/p)$ satisfies (9) with $\eta = 2$ when $c = O(\mathrm{poly}(\log p))$ under the best possible scaling for $J_{\min}$ subject to the walk-summability constraint in (7). Similarly, the small-world ensemble $\mathcal{G}_{\mathrm{Watts}}(p, d, c/p)$ satisfies (9) with $\eta = d + 2$, when $d = O(1)$ and $c = O(\mathrm{poly}(\log p))$.

**Ising Models:** For Ising models, the best possible scaling of the minimum edge potential $J_{\min}$ is when $J_{\min} = \Theta(J^*)$, for the threshold $J^*$ defined in (8). For the ensemble of graphs $\mathcal{G}_{\mathrm{Deg,Girth}}(p; \Delta, g)$ with degree $\Delta$ and girth $g$, we can establish that $J^* = \Theta(1/\Delta)$. When the minimum edge potential achieves the threshold, i.e., $J_{\min} = \Theta(1/\Delta)$, we end up with a similar requirement as in (10) for Gaussian models. Similarly, for both the Erdős-Rényi graph ensemble $\mathcal{G}_{\mathrm{ER}}(p, c/p)$ and small-world ensemble $\mathcal{G}_{\mathrm{Watts}}(p, d, c/p)$, we can establish that the threshold $J^* = \Theta(1/c)$, and thus, the observations made for the Gaussian setting hold for the Ising model as well.

### 3.2 Conditional Mutual Information Threshold Test

Our structure learning procedure is known as the Conditional Mutual Information Threshold Test (CMIT). Let $\mathrm{CMIT}(\mathbf{x}^n; \xi_{n,p}, \eta)$ be the output edge set from CMIT given $n$ i.i.d. samples $\mathbf{x}^n$, a threshold $\xi_{n,p}$ and a constant $\eta \in \mathbb{N}$. The conditional mutual information test proceeds as follows: one computes the empirical conditional mutual information[7] for each node pair $(i,j) \in V^2$ and finds the conditioning set which achieves the minimum, over all subsets of cardinality at most $\eta$,

$$\min_{S \subset V \setminus \{i,j\}, |S| \leq \eta} \widehat{I}(X_i; X_j | \mathbf{X}_S), \tag{11}$$

where $\widehat{I}(X_i; X_j | \mathbf{X}_S)$ denotes the empirical conditional mutual information of $X_i$ and $X_j$ given $\mathbf{X}_S$. If the above minimum value exceeds the given threshold $\xi_{n,p}$, then the node pair is declared as an edge. Recall that the conditional mutual information $I(X_i; X_j | \mathbf{X}_S) = 0$ iff given $\mathbf{X}_S$, the random variables $X_i$ and $X_j$ are conditionally independent.

Thus, (11) seeks to identify non-neighbors, i.e., node pairs which can be separated in the unknown graph $G$. However, since we constrain the conditioning set $|S| \leq \eta$ in (11), the optimal conditioning set may not form an exact separator. Despite this restriction, we establish that the above test can correctly classify the edges and non-neighbors using a suitable threshold $\xi_{n,p}$ subject to the assumptions (A1)–(A4). The threshold $\xi_{n,p}$ is chosen as a function of the number of nodes $p$, the number of samples $n$, and the minimum edge potential $J_{\min}$ as follows:

$$\xi_{n,p} = O(J_{\min}^2), \ \xi_{n,p} = \omega(\alpha^{2\gamma}), \ \xi_{n,p} = \Omega\left(\frac{\log p}{n}\right), \tag{12}$$

where $\gamma$ is the path-threshold in (5) for $(\eta, \gamma)$-local separation to hold and $\alpha$ is given by (7) and (8). The computational complexity of the CMIT algorithm is $O(p^{\eta+2})$. Thus the algorithm is computationally efficient for small $\eta$. Moreover, the algorithm only uses statistics of order $\eta + 2$ in contrast to the convex-relaxation approaches [4–6] which typically use higher-order statistics.

**Theorem 1 (Structural consistency of** CMIT**)** *Assume that (A1)-(A4) hold. Given a Gaussian graphical model or an Ising model Markov on a graph $G_p \sim \mathcal{G}(p; \eta, \gamma)$,* CMIT$(\mathbf{x}^n; \xi_{n,p}, \eta)$ *is structurally consistent. In other words,*

$$\lim_{n,p\to\infty} P\left[\text{CMIT}\left(\{\mathbf{x}^n\}; \xi_{n,p}, \eta\right) \neq G_p\right] = 0. \tag{13}$$

**Consistency guarantee**   The CMIT algorithm consistently recovers the structure of the graphical models with probability tending to one and the probability measure in (4) is with respect to both the graph and the samples.

**Sample-complexity**   The sample complexity of the CMIT scales as $\Omega(J_{\min}^{-4} \log p)$ and is favorable when the minimum edge potential $J_{\min}$ is large. This is intuitive since the edges have stronger potentials when $J_{\min}$ is large. On the other hand, $J_{\min}$ cannot be arbitrarily large due to the assumption (A2). The minimum sample complexity is attained when $J_{\min}$ achieves this upper bound.

It can be established that for both Gaussian and Ising models Markov on a degree-bounded graph ensemble $\mathcal{G}_{\text{Deg}}(p, \Delta)$ with maximum degree $\Delta$ and satisfying assumption (A3), the minimum sample complexity is given by $n = \Omega(\Delta^4 \log p)$ i.e., when $J_{\min} = \Theta(1/\Delta)$.

We can have improved guarantees for the Erdős-Rényi random graphs $\mathcal{G}_{\text{ER}}(p, c/p)$. In the Gaussian setting, the minimum sample complexity can be improved to $n = \Omega(\Delta^2 \log p)$, i.e., when $J_{\min} = \Theta(1/\sqrt{\Delta})$ where the maximum degree scales as $\Delta = \Theta(\log p \log c)$ [7].

On the other hand, for Ising models, the minimum sample complexity can be further improved to $n = \Omega(c^4 \log p)$, i.e., when $J_{\min} = \Theta(J^*) = \Theta(1/c)$. Note that $c/2$ is the expected degree of the $\mathcal{G}_{\text{ER}}(p, c/p)$ ensemble. Specifically, when the Erdős-Rényi random graphs have a bounded average degree $(c = O(1))$, we can obtain a minimum sample complexity of $n = \Omega(\log p)$ for structure estimation of Ising models. Recall that the sample complexity of learning tree models is $\Omega(\log p)$ [23]. Thus, the complexity of learning sparse Erdős-Rényi random graphs is akin to learning trees in certain parameter regimes.

The sample complexity of structure estimation can be improved to $n = \Omega(J_{\min}^{-2} \log p)$ by employing empirical conditional covariances for Gaussian models and empirical conditional variation distances in place of empirical conditional mutual information. However, to present a unified framework for Gaussian and Ising models, we present the CMIT here. See [21, 22] for details.

**Comparison with convex-relaxation approaches**   We now compare our approach for structure learning with convex-relaxation methods. The work by Ravikumar et al. in [5] employs an $\ell_1$-penalized likelihood estimator and under the so-called incoherence conditions, the sample complexity is $n = \Omega((\Delta^2 + J_{\min}^{-2}) \log p)$. Our sample complexity (using conditional covariances) $n = \Omega(J_{\min}^{-2} \log p)$ is the same in terms of $J_{\min}$, while there is no explicit dependence on the maximum degree $\Delta$. Similarly, we match the neighborhood-based regression method by Meinshausen and Buhlmann in [24] under more transparent conditions.

For structure estimation of Ising models, the work in [6] considers $\ell_1$-penalized logistic regression which has a sample complexity of $n = \Omega(\Delta^3 \log p)$ for a degree-bounded ensemble $\mathcal{G}_{\text{Deg}}(p, \Delta)$ satisfying certain "incoherence" conditions. The sample complexity of CMIT, given by $n = \Omega(\Delta^4 \log p)$, is slightly worse, while the modified algorithm described previously has a sample complexity of $n = \Omega(\Delta^2 \log p)$, for general degree-bounded ensembles. Additionally, under the CMIT algorithm, we can guarantee an improved sample complexity of $n = \Omega(c^4 \log p)$ for Erdős-Rényi

random graphs $\mathcal{G}_{\text{ER}}(p, c/p)$ and small-world graphs $\mathcal{G}_{\text{Watts}}(p, d, c/p)$, since the average degree $c/2$ is typically much smaller than the maximum degree $\Delta$. Moreover, note that, the incoherence conditions stated in [6] are NP-hard to establish for general models since they involve the partition function of the model. In contrast, our conditions are transparent and relate to the statistical-physical properties of the model. Moreover, our algorithm is local and requires only low-order statistics, while the method in [6] requires full-order statistics.

**Proof Outline** We first analyze the scenario when exact statistics are available. (i) We establish that for any two non-neighbors $(i, j) \notin G$, the minimum conditional mutual information in (11) (based on exact statistics) does not exceed the threshold $\xi_{n,p}$. (ii) Similarly, we also establish that the conditional mutual information in (11) exceeds the threshold $\xi_{n,p}$ for all neighbors $(i, j) \in G$. (iii) We then extend these results to empirical versions using concentration bounds. See [21, 22] for details.

The main challenge in our proof is step (i). To this end, we analyze the conditional mutual information when the conditioning set is a local separator between $i$ and $j$ and establish that it decays as $p \to \infty$. The techniques involved to establish this for Ising and Gaussian models are different: for Ising models, we employ the self-avoiding walk (SAW) tree construction [25]. For Gaussian models, we use the techniques from walk-sum analysis [14].

## 4 Necessary Conditions for Model Selection

In the previous sections, we proposed and analyzed efficient algorithms for learning the structure of graphical models. We now derive the *necessary* conditions for consistent structure learning. We focus on the ensemble of Erdős-Rényi graphs $\mathcal{G}_{\text{ER}}(p, c/p)$.

For the class of degree-bounded graphs $\mathcal{G}_{\text{Deg}}(p, \Delta)$, necessary conditions on sample complexity have been characterized previously [26] by considering a certain (restricted) set of ensembles. However, a naïve application of such bounds (based on Fano's inequality [27, Ch. 2]) turns out to be too weak for the class of Erdős-Rényi graphs $\mathcal{G}_{\text{ER}}(p, c/p)$. We provide novel necessary conditions for structure learning of Erdős-Rényi graphs. Our techniques may also be applicable to other classes of random graphs.

Recall that a graph $G$ is drawn from the ensemble of Erdős-Rényi graphs $\mathcal{G}_{\text{ER}}(p, c/p)$. Given $n$ i.i.d. samples $\mathbf{X}^n := (\mathbf{X}_1, \ldots, \mathbf{X}_n) \in (\mathcal{X}^p)^n$, the task is to estimate $G$ from $\mathbf{X}^n$. Denote the estimated graph as $\widehat{G} := \widehat{G}(\mathbf{X}^n)$. It is desired to derive tight necessary conditions on the number of samples $n$ (as a function of average degree $c/2$ and number of nodes $p$) so that the *probability of error* $P_e^{(p)} := P(\widehat{G}(\mathbf{X}^n) \neq G) \to 0$ as the number of nodes $p$ tends to infinity. Again, note that the probability measure $P$ is with respect to both the Erdős-Rényi graph and the samples.

**Discrete Graphical Models** Let $H_{\text{b}}(q) := -q \log_2 q - (1 - q) \log_2(1 - q)$ be the binary entropy function. For the Ising model, or more generally any discrete model where each random variable $X_i \in \mathcal{X} = \{1, \ldots, |\mathcal{X}|\}$, we can demonstrate the following:

**Theorem 2 (Weak Converse for Discrete Models)** *For a discrete graphical model Markov on $G \sim \mathcal{G}_{\text{ER}}(p, c/p)$, if $P_e^{(p)} \to 0$, it is necessary for $n$ to satisfy*

$$n \geq \frac{1}{p \log_2 |\mathcal{X}|} \binom{p}{2} H_{\text{b}} \left( \frac{c}{p} \right) \geq \frac{c \log_2 p}{2 \log_2 |\mathcal{X}|}. \tag{14}$$

The above bound does not involve any asymptotic notation and shows transparently, how $n$ has to depend on $p, c$ and $|\mathcal{X}|$ for consistent structure learning. Note that if the cardinality of the random variables $|\mathcal{X}|$ is large, then the necessary sample complexity is small, which makes intuitive sense from a source-coding perspective. Moreover, the above bound states that more samples are required as the average degree $c/2$ increases. Our bound involves only the average degree $c/2$ and not the maximum degree of the graph, which is typically much larger than $c$ [7].

**Gaussian Graphical Models** We now turn out attention to the Gaussian analogue of Theorem 2 under a similar setup. We assume that the $\alpha$-walk-summability condition in assumption (A2a) holds. We are then able to demonstrate the following:

**Theorem 3 (Weak Converse for Gaussian Models)** *For an $\alpha$-walk summable Gaussian graphical model Markov on $G \sim \mathcal{G}_{\mathrm{ER}}(p, c/p)$ as $p \to \infty$, if $P_e^{(p)} \to 0$, we require*

$$n \geq \frac{2}{p \log_2 \left[ 2\pi e \left( \frac{1}{1-\alpha} + 1 \right) \right]} \binom{p}{2} H_{\mathrm{b}} \left( \frac{c}{p} \right) \geq \frac{c \log_2 p}{\log_2 \left[ 2\pi e \left( \frac{1}{1-\alpha} + 1 \right) \right]}. \tag{15}$$

As with Theorem 2, the above bound does not involve any asymptotic notation and similar intuitions hold as before. There is a natural logarithmic dependence on $p$ and a linear dependence on the average degree parameter $c$. Finally, the dependence on $\alpha$ can be explained as follows: any $\alpha$-walk-summable model is also $\beta$-walk-summable for all $\beta > \alpha$. Thus, the class of $\beta$-walk-summable models contains the class of $\alpha$-walk-summable models. This results in a looser bound in (15) for large $\alpha$.

**Analysis tools** Our analysis tools are information-theoretic in nature. A common tool to derive necessary conditions is to resort to Fano's inequality [27, Ch. 2], which (lower) bounds the probability of error $P_e^{(p)}$ as a function of the *conditional entropy* $H(G|\mathbf{X}^n)$ and the size of the set of all graphs with $p$ nodes. However, a naïve application of Fano's inequality results in a trivial lower bound as the set of all graphs, which can be realized by $\mathcal{G}_{\mathrm{ER}}(p, c/p)$ is "too large".

To ameliorate this problem, we focus our attention on the *typical* graphs for applying Fano's inequality and not all graphs. The set of typical graphs has a small cardinality but high probability when $p$ is large. The novelty of our proof lies in our use of both typicality as well as Fano's inequality to derive necessary conditions for structure learning. We can show that (i) the probability of the typical set tends to one as $p \to \infty$, (ii) the graphs in the typical set are almost uniformly distributed (the asymptotic equipartition property), (iii) the cardinality of the typical set is small relative to the set of all graphs. These properties are used to prove Theorems 2 and 3.

## 5   Conclusion

In this paper, we adopted a novel and a unified paradigm for graphical model selection. We presented a simple local algorithm for structure estimation with low computational and sample complexities under a set of mild and transparent conditions. This algorithm succeeds on a wide range of graph ensembles such as the Erdős-Rényi ensemble, small-world networks etc. We also employed novel information-theoretic techniques for establishing necessary conditions for graphical model selection.

**Acknowledgement**

The first author is supported by the setup funds at UCI and in part by the AFOSR under Grant FA9550-10-1-0310, the second author is supported by A*STAR, Singapore and the third author is supported in part by AFOSR under Grant FA9550-08-1-1080.

## Footnotes

[1]A set $S \subset V$ is a separator of sets $A$ and $B$ if the removal of nodes in $S$ separates $A$ and $B$ into distinct components.

[2] Note that the term a.a.s. does not apply to deterministic graph ensembles $\mathcal{G}(p)$ where no randomness is assumed, and in this setting, we assume that the property $\mathcal{Q}$ holds for every graph in the ensemble.

[3] A minimal separator is a separator of smallest cardinality.

[4]Two cycles are said to overlap if they have common vertices.

[5]The notations $\omega(\cdot), \Omega(\cdot), o(\cdot)$ and $O(\cdot)$ refer to asymptotics as the number of variables $p \to \infty$.

[6]We say that two sequences $f(p), g(p)$ satisfy $f(p) = \widetilde{\omega}(g(p))$, if $\frac{f(p)}{g(p)\log p} \to \infty$ as $p \to \infty$.

[7]The empirical conditional mutual information is obtained by first computing the empirical distribution and then computing its conditional mutual information.

## References

[1] S. Lauritzen, *Graphical models: Clarendon Press*.   Clarendon Press, 1996.

[2] D. Karger and N. Srebro, "Learning Markov Networks: Maximum Bounded Tree-width Graphs," in *Proc. of ACM-SIAM symposium on Discrete algorithms*, 2001, pp. 392–401.

[3] C. Chow and C. Liu, "Approximating Discrete Probability Distributions with Dependence Trees," *IEEE Tran. on Information Theory*, vol. 14, no. 3, pp. 462–467, 1968.

[4] A. d'Aspremont, O. Banerjee, and L. El Ghaoui, "First-order methods for sparse covariance selection," *SIAM. J. Matrix Anal. & Appl.*, vol. 30, no. 56, 2008.

[5] P. Ravikumar, M. Wainwright, G. Raskutti, and B. Yu, "High-dimensional covariance estimation by minimizing $\ell_1$-penalized log-determinant divergence," *Arxiv preprint arXiv:0811.3628*, 2008.

[6] P. Ravikumar, M. Wainwright, and J. Lafferty, "High-dimensional Ising Model Selection Using l1-Regularized Logistic Regression," *Annals of Statistics*, 2008.

[7] B. Bollobás, *Random Graphs*.   Academic Press, 1985.

[8] F. Chung and L. Lu, *Complex graphs and network*.   Amer. Mathematical Society, 2006.

[9] D. Watts and S. Strogatz, "Collective dynamics of small-worldnetworks," *Nature*, vol. 393, no. 6684, pp. 440–442, 1998.

[10] M. Newman, D. Watts, and S. Strogatz, "Random graph models of social networks," *Proc. of the National Academy of Sciences of the United States of America*, vol. 99, no. Suppl 1, 2002.

[11] R. Albert and A. Barabási, "Statistical mechanics of complex networks," *Reviews of modern physics*, vol. 74, no. 1, p. 47, 2002.

[12] H. Georgii, *Gibbs Measures and Phase Transitions*.   Walter de Gruyter, 1988.

[13] J. Bento and A. Montanari, "Which Graphical Models are Difficult to Learn?" in *Proc. of Neural Information Processing Systems (NIPS)*, Vancouver, Canada, Dec. 2009.

[14] D. Malioutov, J. Johnson, and A. Willsky, "Walk-Sums and Belief Propagation in Gaussian Graphical Models," *J. of Machine Learning Research*, vol. 7, pp. 2031–2064, 2006.

[15] G. Bresler, E. Mossel, and A. Sly, "Reconstruction of Markov Random Fields from Samples: Some Observations and Algorithms," in *Intl. workshop APPROX Approximation, Randomization and Combinatorial Optimization*. Springer, 2008, pp. 343–356.

[16] P. Netrapalli, S. Banerjee, S. Sanghavi, and S. Shakkottai, "Greedy Learning of Markov Network Structure ," in *Proc. of Allerton Conf. on Communication, Control and Computing*, Monticello, USA, Sept. 2010.

[17] F. Chung, *Spectral graph theory*.   Amer Mathematical Society, 1997.

[18] A. Gamburd, S. Hoory, M. Shahshahani, A. Shalev, and B. Virag, "On the girth of random cayley graphs," *Random Structures & Algorithms*, vol. 35, no. 1, pp. 100–117, 2009.

[19] S. Dommers, C. Giardinà, and R. van der Hofstad, "Ising models on power-law random graphs," *Journal of Statistical Physics*, pp. 1–23, 2010.

[20] B. McKay, N. Wormald, and B. Wysocka, "Short cycles in random regular graphs," *The Electronic Journal of Combinatorics*, vol. 11, no. R66, p. 1, 2004.

[21] A. Anandkumar, V. Y. F. Tan, and A. S. Willsky, "High-Dimensional Structure Learning of Ising Models: Tractable Graph Families," *Preprint, Available on ArXiv 1107.1736*, June 2011.

[22] ——, "High-Dimensional Gaussian Graphical Model Selection: Tractable Graph Families," *Preprint, ArXiv 1107.1270*, June 2011.

[23] V. Tan, A. Anandkumar, and A. Willsky, "Learning Markov Forest Models: Analysis of Error Rates," *J. of Machine Learning Research*, vol. 12, pp. 1617–1653, May 2011.

[24] N. Meinshausen and P. Buehlmann, "High Dimensional Graphs and Variable Selection With the Lasso," *Annals of Statistics*, vol. 34, no. 3, pp. 1436–1462, 2006.

[25] D. Weitz, "Counting independent sets up to the tree threshold," in *Proc. of ACM symp. on Theory of computing*, 2006, pp. 140–149.

[26] W. Wang, M. Wainwright, and K. Ramchandran, "Information-theoretic bounds on model selection for Gaussian Markov random fields," in *IEEE International Symposium on Information Theory Proceedings (ISIT)*, Austin, Tx, June 2010.

[27] T. Cover and J. Thomas, *Elements of Information Theory*.   John Wiley & Sons, Inc., 2006.

